# Adjoint-Functions and Temporal Learning Algorithms in Neural Networks

N. Toomarian and J. Barhen
Jet Propulsion Laboratory
California Institute of Technology
Pasadena, CA 91109

## Abstract

The development of learning algorithms is generally based upon the minimization of an energy function. It is a fundamental requirement to compute the gradient of this energy function with respect to the various parameters of the neural architecture, e.g., synaptic weights, neural gain,etc. In principle, this requires solving a system of nonlinear equations for each parameter of the model, which is computationally very expensive. A new methodology for neural learning of time-dependent nonlinear mappings is presented. It exploits the concept of adjoint operators to enable a fast global computation of the network's response to perturbations in all the systems parameters. The importance of the time boundary conditions of the adjoint functions is discussed. An algorithm is presented in which the adjoint sensitivity equations are solved *simultaneously* (i.e., forward in time) along with the nonlinear dynamics of the neural networks. This methodology makes real-time applications and hardware implementation of temporal learning feasible.

## 1 INTRODUCTION

Early efforts in the area of training artificial neural networks have largely focused on the study of schemes for encoding nonlinear mapping characterized by time-independent inputs and outputs. The most widely used approach in this context has been the error backpropagation algorithm (Werbos, 1974), which involves either static i.e., "feedforward" (Rumelhart, 1986), or dynamic i.e., "recurrent" ( Pineda, 1988) networks. In this context ( Barhen et al, 1989, 1990a, 1990b), have exploited

the concepts of adjoint operators and terminal attractors. These concepts provide a firm mathematical foundation for learning such mappings with dynamical neural networks, while achieving a considerable reduction in the overall computational costs (Barhen et al, 1991).

Recently, there has been a wide interest in developing learning algorithms capable of modeling time-dependent phenomena ( Hirsh, 1989). In a more restricted application oriented, domain attention has focused on learning temporal sequences. The problem can be formulated as minimization, over an arbitrary but finite time interval, of an appropriate error functional. Thus, the gradients of the functional with respect to the various parameters of the neural architecture, e.g., synaptic weights, neural gains, etc. must be computed.

A number of methods have been proposed for carrying out this task, a recent survey of which can be found in (Pearlmutter, 1990). Here, we will briefly mention only those which are relevant to our work. Williams and Zipser(1989) discuss a scheme similar to the well known "Forward Sensitivity Equations" of sensitivity theory (Cacuci, 1981 and Toomarian et al, 1987), in which the same set of sensitivity equations has to be solved again and again for each network parameter of interest. Clearly, this is computationally very expensive, and scales poorly to large systems. Pearlmutter (1989), on the other hand, describes a variational approach which yields a set of equations which are similar to the "Adjoint Sensitivity Equations" (Cacuci, 1981 and Toomarian et al, 1987). These equations must be solved backwards in time and involve storage of the state variables from the activation network dynamics, which is impractical. These authors ( Toomarian and Barhen, 1990 ) have suggested a new method which, in contradistinction to previous approaches, solves the adjoint system of equations forward in time, concomitantly with the neural activation dynamics. A potential drawback of this method lies in the fact that these adjoint equations have to be treated in terms of distributions which precludes straight-forward numerical implementation. Finally, Pineda (1990), suggests combining the existence of disparate time scales with a heuristic gradient computation. However, the underlying adiabatic assumptions and highly "approximate" gradient evaluation technique place severe limits on the applicability of his approach.

In this paper we introduce a rigorous derivation of two novel systems of adjoint equations, which can be solved *simultaneously* (i.e., forward in time) with the network dynamics, and thereby enable the implementation of temporal learning algorithms in a computationally efficient manner. Numerical simulations and comparison with previously available results will be presented elsewhere( Toomarian and Barhen, 1991).

## 2   TEMPORAL LEARNING

We formalize a neural network as an adaptive dynamical system whose temporal evolution is governed by the following set of coupled nonlinear differential equations:

$$\dot{u}_n + \kappa_n u_n = g_n[\gamma_n(\sum_m T_{nm} u_m + I_n)] \quad t > 0 \quad (1)$$

where $u_n$ represents the output of the $n$th neuron [$u_n(0)$ being the initial state], and $T_{nm}$ denotes the synaptic coupling from the $m-$th to the $n-$th neuron. The constant $\kappa_n$ characterizes the decay of neuron activity. The sigmoidal function $g(\cdot)$ modulates the neural response, with gain given by $\gamma$; typically, $g(\gamma x) = \tanh(\gamma x)$. The time-dependent "source" term, $I_n(t)$, encodes component-contribution of the target temporal pattern $a(t)$ via the expression

$$I_n(t) = \begin{cases} a_n(t) & \text{if } n \in S_X \\ 0 & \text{if } n \in S_H \cup S_Y \end{cases} \tag{2}$$

The topographic input, output, and hidden network partitions $S_X$, $S_Y$ and $S_H$, respectively, are architectural requirements related to the encoding of mapping-type problems. Details are given in Barhen et al (1989).

To proceed formally with the development of a temporal learning algorithm, we consider an approach based upon the minimization of a "neuromorphic" energy functional $E$, given by the following expression

$$E(\bar{u}, \bar{p}) = \int_t \frac{1}{2} \sum_n \Gamma_n^2 \, dt = \int_t F \, dt \tag{3}$$

where

$$\Gamma_n(t) = \begin{cases} a_n(t) - u_n(t) & \text{if } n \in S_Y \\ 0 & \text{if } n \in S_X \cup S_H \end{cases} \tag{4}$$

In our model the internal dynamical parameters of interest are the synaptic strengths $T_{nm}$ of the interconnection topology, the characteristic decay constants $\kappa_n$, and the gain parameters $\gamma_n$. Therefore, the vector of system parameters ( Barhen et al, 1990b) should be

$$\bar{p} = \{| \ T_{11}, \cdots, T_{NN} \ | \ \kappa_1, \cdots, \kappa_N \ | \ \gamma_1, \cdots, \gamma_N \ |\} \tag{5a}$$

In this paper, however, for illustration purposes and simplicity, we will limit ourselves in terms of parameters to the synaptic interconnections only. Hence, the vector of system parameters will have $M = N^2$ elements

$$\bar{p} = \{ \ T_{11}, \cdots, T_{NN}\} \tag{5b}$$

We will assume that elements of $\bar{p}$ are, in principle, independent. Furthermore, we will also assume that, for a specific choice of parameters and set of initial conditions, a unique solution of Eq. (1) exists. Hence, $\bar{u}$ is an implicit function of $\bar{p}$.

Lyapunov stability requires the energy functional to be monotonically decreasing during learning time, $\tau$. This translates into

$$\frac{dE}{d\tau} = \sum_{\mu=1}^{M} \frac{dE}{dp_\mu} \cdot \frac{dp_\mu}{d\tau} < 0 \tag{6}$$

Thus, one can always choose, with $\eta > 0$

$$\frac{dp_\mu}{d\tau} = -\eta \frac{dE}{dp_\mu} \tag{7}$$

Integrating the above dynamical system over the interval $[\tau, \tau + \Delta\tau]$, one obtains,

$$p_\mu(\tau + \Delta\tau) = p_\mu(\tau) - \eta \int_\tau^{\tau+\Delta\tau} \frac{dE}{dp_\mu} d\tau \tag{8}$$

Equation (8) implies that, in order to update a system parameter $p_\mu$, one must evaluate the gradient of $E$ with respect to $p_\mu$ in the interval $[\tau, \tau+\Delta\tau]$. Furthermore, using Eq. (3) and observing that the time integral and derivative with respect to $p_\mu$, permute one can write;

$$\frac{dE}{dp_\mu} = \int_t \frac{dF}{dp_\mu} \, dt = \int_t \frac{\partial F}{\partial p_\mu} \, dt + \int_t \frac{\partial F}{\partial \bar{u}} \cdot \frac{\partial \bar{u}}{\partial p_\mu} \, dt \tag{9}$$

Since $F$ is known analytically [viz. Eq. (3)] computation of $\partial F/\partial u_n$ and $\partial F/\partial p_\mu$ is straightforward.

$$\frac{\partial F}{\partial u_n} = -\Gamma_n \tag{10a}$$

$$\frac{\partial F}{\partial p_\mu} = 0 \tag{10b}$$

Thus, the quantity that needs to be determined is the vector $\partial \bar{u}/\partial p_\mu$. Differentiating the activation dynamics, Eq. (1), with respect to $p_\mu$, we observe that the time derivative and partial derivative with respect to $p_\mu$ commute. Using the shorthand notation $\partial(\cdots)/\partial p_\mu = (\cdots)_{,\mu}$ we obtain a set of equations to be referred to as "Forward Sensitivity Equations-FSE":

$$\begin{cases} \dot{u}_{n,\mu} + \sum_m A_{nm}\, u_{m,\mu} = S_{n,\mu} & t > 0 \\ u_{n,\mu} = 0 & t = 0 \end{cases} \tag{12}$$

in which

$$A_{nm} = \kappa_n\, \delta_{nm} - \gamma_n\, \hat{g}_n\, T_{nm} \tag{13}$$

$$S_{n,\mu} = \gamma_n\, \hat{g}_n \sum_m T_{nm}\, u_m\, \delta_{p_\mu, T_{nm}} \tag{14}$$

where $\hat{g}_n$ represents the derivative of $g_n$ with respect to $u_n$, and $\delta$ denotes the Kronecker symbol. Since the initial conditions of the activation dynamics, Eq.(1), are excluded from the system parameter vector $\bar{p}$, the initial conditions of the forward sensitivity equations will be taken as zero. Computation of the gradients, via Eq. (9), using the forward sensitivity scheme as proposed by William and Zipser (1989), would require solving Eq. (12), $N^2$ times, since the source term explicitly depends on $p_\mu$. The system of equations (12) has $N$ equations, each of which requires summation over all $N$ neurons. Hence, the amount of computation ( measured in multiply-accumulates, scales like $N^4$ per time step. We assume that the interval between $t_0$ to $t_f$ is divided to $L$ time steps. Therefore, the total number of multiply-accumulates scales like $N^4 L$. Clearly, the scaling properties of this approach are very poor and it can not be practically applied to very large networks. On the other hand, this method has also inherent advantages. The FSE are solved forward in time along with the nonlinear dynamics of the neural networks. Therefore, there is no need for or a large amount of memory. Since $u_{n,\mu}$ has $N^3$ components, that is all needed to be stored.

In order to reduce the computational costs, an alternative approach can be considered. It is based upon the concept of adjoint operators, and eliminates the need for explicit appearance of $\bar{u}_{,\mu}$ in Eq. (9). A vector of adjoint functions, $\bar{v}$ is obtained, which contain all the information required for computing all the "sensitivities", $dE/dp_\mu$. The necessary and sufficient conditions for constructing adjoint equations are discussed elsewhere ( Toomarian et al, 1987 and references therein).

It can be shown that an Adjoint System of Equations-ASE, pertaining to the forward system of equations (12), can be formally written as

$$-\dot{v}_n + \sum_m A^T_{nm}\, v_m = S^*_n \qquad t > 0 \qquad (15)$$

In order to specify Eq. (15) in closed mathematical form , we must define the source term $S^*_n$ and time- boundary conditions for the system. Both should be independent of $p_\mu$ and its derivatives.

By identifying $S^*_n$ with $\partial F / \partial u_n$ and selecting the final time condition $\bar{v}(t = t_f) = 0$, a system of equations is obtained, which is similar to those proposed by Pearlmutter. The method requires that the neural activation dynamics, i.e., Eq. (1), be solved first forward in time, as followed by the ASE, Eq. (15), integrated backwards in time. The computation requirement of this approach scales as $N^2 L$. However, a major drawback to date has resided with the necessity to store quantities such as $\hat{\bar{g}}$, $\bar{S}^*$ and $\bar{S}_{,\mu}$ at each time step. Thus, the memory requirements for this method scale as $N^2 L$.

By selecting $\bar{S}^* = \frac{\partial F}{\partial \bar{u}} - \bar{v}\delta(t - t_f)$ and initial conditions $\bar{v}(t = 0) = 0$, these authors ( Toomarian and Barhen 1990 ) have suggested a method which, in contradistinction to previous approaches, enables the ASE to be integrated forward in time, i.e., concomitantly with the neural activation dynamics. This approach saves a large amount of storage, which scales only as $N^2$. The computation complexity of this method, is similar to that of backward integration and scales as $N^2 L$. A potential drawback lies in the fact that Eq. (15) must then be treated in terms of distributions, which precludes straightforward numerical implementation.

At this stage, we introduce a new paradigm which will enable us to evolve the adjoint dynamics, Eq. (15) forward in time, but without the difficulties associated with solutions in the sense of distributions. We multiply the FSE, Eq. (12), by $\bar{v}$ and the ASE, Eq. (15), by $\bar{u}_{,\mu}$, subtract the two resulting equations and integrate over the time interval $(t_o, t_f)$. This procedure yields the bilinear form:

$$(\bar{v}\,\bar{u}_{,\mu})_{t_f} - (\bar{v}\,\bar{u}_{,\mu})_{t_o} = \int_{t_o}^{t_f} [(\bar{v}\,\bar{S}_{,\mu}) - (\bar{u}_{,\mu}\,\bar{S}^*)]dt \qquad (16)$$

To proceed, we select

$$\begin{cases} \bar{S}^* = \frac{\partial F}{\partial \bar{u}} \\ \bar{v}(t = 0) = 0. \end{cases} \qquad (17)$$

Thus, Eq. (16) can be rewritten as:

$$\int_t \frac{\partial F}{\partial \bar{u}}\bar{u}_{,\mu}dt \equiv \int_t \bar{S}^*\,\bar{u}_{,\mu}dt = \int_t \bar{v}\,\bar{S}_{,\mu}dt - [\bar{v}\,\bar{u}_{,\mu}]_{t_f} \qquad (18)$$

The first term in the RHS of Eq. (18) can be computed by using the values of $\bar{v}$ obtained by solving the ASE, (Eqs. (15) and (17)), forward in time. The main difficulty resides in the evaluation of the second term in the RHS of Eq. (18), i.e., $[\bar{v}\,\bar{u}_{,\mu}]_{t_f}$. To compute it, we now introduce an auxiliary adjoint system:

$$-\dot{z}_n + \sum_m A^T_{nm}\, z_m = \hat{S}_n \qquad t > 0 \tag{19}$$

in which we select

$$\begin{cases} \hat{\bar{S}} = \bar{v}(t)\delta(t - t_f) \\ \bar{z}(t_f) = 0. \end{cases} \tag{20}$$

Note that, eventhough we selected $\bar{z}(t_f) = 0$, we are also interested in solving this auxiliary adjoint system forward in time. Thus, the critical issue is how to select the initial condition (i.e. $\bar{z}(t_o)$), that would result in $\bar{z}(t_f) = 0$. The bilinear form associated with the dynamical systems $\bar{u}_{,\mu}$ and $\bar{z}$ can be derived in a similar fashion to Eq. (16). Its expression is:

$$(\bar{z}\,\bar{u}_{,\mu})_{t_f} - (\bar{z}\,\bar{u}_{,\mu})_{t_o} = \int_{t_o}^{t_f} [(\bar{z}\,\bar{S}_{,\mu}) - (\bar{u}_{,\mu}\,\hat{S})]dt \tag{21}$$

Incorporating $\hat{\bar{S}}, \bar{z}(t_f)$ and the initial condition of Eq. (12) into Eq. (21), we obtain;

$$\int_{t_o}^{t_f} (\bar{u}_{,\mu}\,\hat{\bar{S}})dt = [\bar{v}\,\bar{u}_{,\mu}]_{t_f} = \int_{t_o}^{t_f} (\bar{z}\,\bar{S}_{,\mu})dt \tag{22}$$

In order to provide a simple illustration on how the problem of selecting the initial conditions for the $\bar{z}$-dynamics can be addressed, we assume, for a moment, that the matrix $A$ in Eq. (19) is time independent. Hence, the formal solution of Eq. (19) can be written as:

$$\bar{z}(t) = \bar{z}(t_o)e^{A^T(t-t_o)} \tag{23a}$$

$$\bar{z}(t_f) = \bar{z}(t_o)e^{A^T(t_f-t_o)} - \bar{v}(t_f) \tag{23b}$$

Therefore, in principle, Eq. (22) can be expressed in terms of $\bar{z}(t_o)$, using Eq. (23a). At time $t_f$, where $\bar{v}(t_f)$ is known from the solution of Eq. (15), one can calculate the vector $\bar{z}(t_o)$, from Eq. (23b), with $\bar{z}(t_f) = 0$.

In the problem under consideration, however, the matrix $A$ in Eq. (19) is time dependent (viz Eq. (13)). Thus the auxiliary adjoint equations will be solved by means of finite differences. Usually, the same numerical scheme that is used for Eqs. (1) and (15) will be adopted. For illustrative purposes, we limit the discussion in the sequel to the first order approximation i.e.;

$$-\frac{(\bar{z}^{l+1} - \bar{z}^l)}{\Delta t} + A^l \bar{z}^l = 0 \qquad 0 < l < L \tag{24}$$

From this equation one can easily show that

$$\bar{z}^{l+1} = B^l \cdot B^{l-1} \cdots B^1 \cdot B^0 \bar{z}(t_o) = B^{l\prime} \bar{z}(t_o) \tag{25}$$

in which

$$B^l = I + \Delta t \, A^l \tag{26}$$

where $I$ is the identity matrix. Thus, the RHS of Eq. (22) can be rewritten as:

$$[\bar{v} \, \bar{u}_{,\mu}]_{t_f} = [\sum_l B^{(l-1)\prime} \bar{S}_{,\mu}] \bar{z}(t_o) \, \Delta t \tag{27}$$

The initial conditions $\bar{z}(t_o)$ can easily be found at time $t_f$, i.e., at iteration stop $L$, by solving the algebraic equation:

$$B^{(L-1)\prime} \bar{z}(t_o) = \bar{v}(t_f) \tag{28}$$

In summary, the computation of the gradients i.e. Eq. (8) involves two stages, corresponding to the two terms in the RHS of Eq. (18). The first term is calculated using the adjoint functions $\bar{v}$ obtained from Eq. (15). The computational complexity is $N^2 L$. The second term is calculated via Eq. (27), and involves two steps: a) kernel propagation, which requires multiplication of two matrices $B^l$ and $B^{(l-1)}$ at each time step; the computational complexity scales as $N^3 L$; b) numerical integration via Eq. (24) which requires a matrix vector multiplication at each time step; hence, it scales as $N^2 L$. Thus, the overall computational complexity of this approach is of the order $N^3 L$. Notice, however, that here the storage needed is minimal and equal to $N^2$.

## 3    CONCLUSIONS

A new methodology for neural learning of time-dependent nonlinear mappings is presented. It exploits the concept of adjoint operators. The resulting algorithm enables computation of the gradient of an energy function with respect to various parameters of the network architecture in a highly efficient manner. Specifically, it combines the advantage of dramatic reductions in computational complexity inherent in adjoint methods with the ability to solve the equations forward in time. Not only is a large amount of computation and storage saved, but the handling of real-time applications becomes also possible. This methodology also makes the hardware implementation of temporal learning attractive.

**Acknowledgments**

This research was carried out at the Center for Space Microelectronics Technology, Jet Propulsion Laboratory, California Institute of Technology. Support for the work came from Agencies of the U.S. Department of Defense including the Naval Weapons Center (China Lake, CA), and from the Office of Basic Energy Sciences of the Department of Energy, through an agreement with the National Aeronautics and Space Administration. The authors acknowledge helpful discussions with J. Martin and D. Andes from Navel Weapons Center.

## References

Barhen, J., Gulati, S., and Zak, M., 1989, "Neural Learning of Constrained Nonlinear Transformations", *IEEE Computer*, 22(6), 67-76.

Barhen, J., Toomarian, N., and Gulati, S., 1990a, " Adjoint Operator Algorithms for Faster Learning in Dynamical Neural Networks", *Adv. Neur. Inf. Proc. Sys.*, 2, 498-508.

Barhen, J., Toomarian, N., and Gulati, S., 1990b, "Application of Adjoint Operators to Neural Learning", *Appl. Math. Lett.*, 3 (3), 13-18.

Barhen, J., Toomarian, N., and Gulati, S., 1991, "Fast Neural Learning Algorithms Using Adjoint Operators", Submitted to *IEEE Trans. of Neural Networks*

Cacuci, D. G., 1981, "Sensitivity Theory for Nonlinear Systems", *J. Math. Phys.*, 22 (12), 2794-2802.

Hirsch, M. W., 1989, "Convergent Activation Dynamics in Continuous Time Networks", *Neural Networks*, 2 (5), 331-349.

Pearlmutter, B. A., 1989, "Learning State Space Trajectories in Recurrent Neural Networks", *Neural Computation*, 1 (2), 263-269.

Pearlmutter, B. A., 1990, "Dynamic Recurrent Neural Networks", Technical Report CMU-CS-90-196, School of Computer Science, Carnegie Mellon University, Pittsburgh, Pa.

Pineda, F., 1988, "Dynamics and Architecture in Neural Computation", *J. of Complexity, 4*, 216-245.

Pineda, F., 1990, "Time Dependent Adaptive Neural Networks", *Adv. Neur. Inf. Proc. Sys.*, 2, 710-718.

Rumelhart, D. E., and McC.and, J. L., 1986, *Parallel and Distributed Processing*, MIT Press.

Toomarian, N., Wacholder, E., and Kaizerman, S., 1987, "Sensitivity Analysis of Two-Phase Flow Problems", *Nucl. Sci. Eng.*, 99 (1), 53-81.

Toomarian, N. and Barhen, J., 1990, "Adjoint Operators and Non- Adiabatic Algorithms in Neural Networks", *Appl. Math. Lett.*, (in press).

Toomarian, N. and Barhen, J., 1991, " Learning a Trajectory Using Adjoint Functions", submitted to *Neural Networks*

Werbos, P., 1974, "Beyond Regression: New Tools for Prediction and Analysis in The Behavioral Sciences", Ph.D. Thesis, Harvard Univ.

Williams, R. J., and Zipser, D., 1989, "A Learning Algorithm for Continually Running Fully Recurrent Neural Networks", *Neural Computation*, 1 (2), 270-280.


